# Finding the $M$ Most Probable Configurations Using Loopy Belief Propagation

**Chen Yanover and Yair Weiss**
School of Computer Science and Engineering
The Hebrew University of Jerusalem
91904 Jerusalem, Israel
*{cheny,yweiss}@cs.huji.ac.il*

## Abstract

Loopy belief propagation (BP) has been successfully used in a number of difficult graphical models to find the most probable configuration of the hidden variables. In applications ranging from protein folding to image analysis one would like to find not just the best configuration but rather the top $M$. While this problem has been solved using the junction tree formalism, in many real world problems the clique size in the junction tree is prohibitively large. In this work we address the problem of finding the $M$ best configurations when exact inference is impossible.

We start by developing a new exact inference algorithm for calculating the best configurations that uses only max-marginals. For approximate inference, we replace the max-marginals with the beliefs calculated using max-product BP and generalized BP. We show empirically that the algorithm can accurately and rapidly approximate the M best configurations in graphs with hundreds of variables.

## 1    Introduction

Considerable progress has been made in the field of approximate inference using techniques such as variational methods [7], Monte-Carlo methods [5], mini-bucket elimination [4] and belief propagation (BP) [6]. These techniques allow approximate solutions to various inference tasks in graphical models where building a junction tree is infeasible due to the exponentially large clique size. The inference tasks that have been considered include calculating marginal probabilities, finding the most likely configuration, and evaluating or bounding the log likelihood.

In this paper we consider an inference task that has not been tackled with the same tools of approximate inference: calculating the $M$ most probable configurations (MPCs). This is a natural task in many applications. As a motivating example, consider the protein folding task known as the side-chain prediction problem. In our previous work [17], we showed how to find the minimal-energy side-chain configuration using approximate inference in a graphical model. The graph has 300

nodes and the clique size in a junction tree calculated using standard software [10] can be up to an order of $10^{42}$, so that exact inference is obviously impossible. We showed that loopy max-product belief propagation (BP) achieved excellent results in finding the first MPC for this graph. In the few cases where BP did not converge, Generalized Belief Propagation (GBP) always converge, with an increase in computation. But we are also interested in finding the second best configuration, the third best or, more generally, the top $M$ configurations. Can this also be done with BP ?

The problem of finding the $M$ MPCs has been successfully solved within the junction tree (JT) framework. However, to the best of our knowledge, there has been no equivalent solution when building a junction tree is infeasible. A simple solution would be outputting the top $M$ configurations that are generated by a Monte-Carlo simulation or by a local search algorithm from multiple initializations. As we show in our simulations, both of these solutions are unsatisfactory. Alternatively, one can attempt to use more sophisticated heuristically guided search methods (such as $A^*$) or use exact MPCs algorithms on an approximated, reduced size junction tree [4, 1]. However, given the success of BP and GBP in finding the first MPC in similar problems [6, 9] it is natural to look for a method based on BP. In this paper we develop such an algorithm. We start by showing why the standard algorithm [11] for calculating the top $M$ MPCs cannot be used in graphs with cycles. We then introduce a novel algorithm called Best Max-Marginal First (BMMF) and show that when the max-marginals are exact it provably finds the $M$ MPCs. We show simulation results of BMMF in graphs where exact inference is impossible, with excellent performance on challenging graphical models with hundreds of variables.

## 2 Exact MPCs algorithms

We assume our hidden variables are denoted by a vector $X$, $N = |X|$ and the observed variables by $Y$, where $Y = y$. Let $m_k = (m_k(1), m_k(2), \cdots, m_k(N))$ denote the $k^{th}$ MPC. We first seek a configuration $m_1$ that maximizes $\Pr(X = x|y)$. Pearl, Dawid and others [12, 3, 11] have shown that this configuration can be calculated using a quantity known as max-marginals (MMs):

$$\mathsf{max\_marginal}(i, j) = \max_{x:x(i)=j} \Pr(X = x|y) \qquad (1)$$

*Max-marginal lemma:* If there exists a unique MAP assignment $m_1$ (i.e. $\Pr(X = m_1|y) > \Pr(X = x|y), \forall x \neq m_1$) then $x_1$ defined by $x_1(i) = \arg\max_j \mathsf{max\_marginal}(i, j)$ will recover the MAP assignment, $m_1 = x_1$.

*Proof:* Suppose, that there exists $i$ for which $m_1(i) = k$, $x_1(i) = l$, and $k \neq l$. It follows that $\max_{x:x(i)=k} \Pr(X = x|y) > \max_{x:x(i)=l} \Pr(X = x|y)$ which is a contradiction to the definition of $x_1$. □

When the graph is a tree, the MMs can be calculated exactly using max-product belief propagation [16, 15, 12] using two passes: one up the tree and the other down the tree. Similarly, for an arbitrary graph they can be calculated exactly using two passes of max-propagation in the junction tree [2, 11, 3].

A more efficient algorithm for calculating $m_1$ requires only one pass of max-propagation. After calculating the max-marginal exactly at the root node, the MAP assignment $m_1$ can be calculated by tracing back the pointers that were used during the max-propagation [11]. Figure 1a illustrates this traceback operation in the Viterbi algorithm in HMMs [13] (the pairwise potentials favor configurations where neighboring nodes have different values). After calculating messages from left

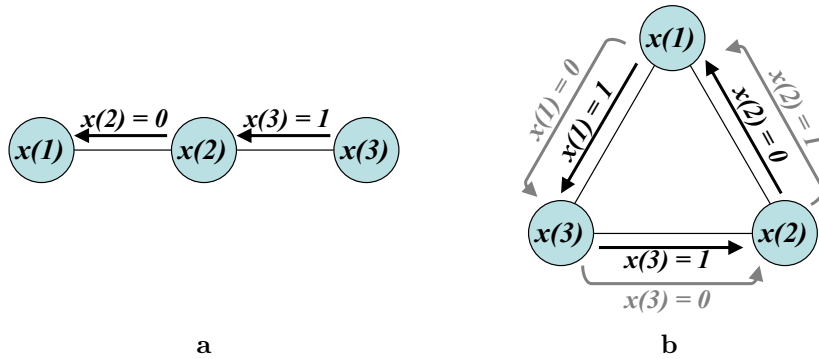

Figure 1: **a.** The traceback operation in the Viterbi algorithm. The MAP configuration can be calculated by a forward message passing scheme followed by a backward "traceback". **b.** The same traceback operation applied to a loopy graph may give inconsistent results.

to right using max-product, we have the max-marginal at node 3 and can calculate $x_1(3) = 1$. We then use the value of $x_1(3)$ and the message from node 1 to 2 to find $x_1(2) = 0$. Similarly, we then trace back to find the value of $x_1(1)$.

These traceback operations, however, are problematic in loopy graphs. Figure 1b shows a simple example from [15] with the same potentials as in figure 1a. After setting $x_1(3) = 1$ we traceback and find $x_1(2) = 0$, $x_1(1) = 1$ and finally $x_1(3) = 0$, which is obviously inconsistent with our initial choice.

One advantage of using traceback is that it can recover $m_1$ even if there are "ties" in the MMs, i.e. when there exists a max-marginal that has a non-unique maximizing value. When there are ties, the max-marginal lemma no longer holds and independently maximizing the MMs will not find $m_1$ (cf. [12]).

Finding $m_1$ using only MMs requires multiple computation of MMs — each time with the additional constraint $x(i) = j$, where $i$ is a tied node and $j$ one of its maximizing values — until no ties exist. It is easy to show that this algorithm will recover $m_1$. The proof is a special case of the proof we present for claim 2 in the next section. However, we need to recalculate the MMs many times until no more ties exist. This is the price we pay for not being able to use traceback. The situation is similar if we seek the $M$ MPCs.

## 2.1 The Simplified Max-Flow Propagation Algorithm

Nilsson's Simplified Max-Flow Propagation (SMFP) [11] starts by calculating the MMs and using the max-marginal lemma to find $m_1$. Since $m_2$ must differ from $m_1$ in at least one variable, the algorithm defines $N$ conditioning sets, $C_i \triangleq (x(1) = m_1(1), x(2) = m_1(2), \cdots, x(i{-}1) = m_1(i{-}1), x(i) \neq m_1(i))$. It then uses the max-marginal lemma to find the most probable configuration given each conditioning set, $x_i = \arg\max_x \Pr(X = x|y, C_i)$ and finally $m_2 = \arg\max_{x \in \{x_i\}} \Pr(X = x|y)$. Since the conditioning sets form a partition, it is easy to show that the algorithm finds $m_2$ after $N$ calculations of the MMs. Similarly, to find $m_k$ the algorithm uses the fact that $m_k$ must differ from $m_1, m_2, \cdots, m_{k-1}$ in at least one variable and forms a new set of up to $N$ conditioning sets. Using the max-marginal lemma one can find the MPC given each of these new conditioning sets. This gives up to $N$ new candidates, in addition to $(k{-}1)(N{-}1)$ previously calculated candidates. The

**t = 1**

| | | x(1) | x(2) | x(3) | x(4) |
|---|---|---|---|---|---|
| $(i_1, j_1, s_1)$ | | | | | |
| CONSTRAINTS$_1$ | | | | | |
| SCORE$_1$ | 0 | .175 | .175 | .3914 | .3914 |
| | 1 | .3914 | .3914 | .1892 | .175 |
| $x_1$ | | 1 | 1 | 0 | 0 |
| USED$_2$ | | | | | |

**t = 2**

| | | x(1) | x(2) | x(3) | x(4) |
|---|---|---|---|---|---|
| $(i_2, j_2, s_2)$ | | (3,1,1) | | | |
| CONSTRAINTS$_2$ | | {x(3)=1} | | | |
| SCORE$_2$ | 0 | .035 | .035 | 0 | .1892 |
| | 1 | .1892 | .1892 | _**.1892**_ | .035 |
| $x_2$ | | 1 | 1 | 1 | 0 |
| USED$_3$ | | {(3,1,1)} | | | |
| CONSTRAINTS$_1$ | | {x(3)≠1} | | | |
| SCORE$_1$ | 0 | .175 | .175 | .3914 | .3914 |
| | 1 | .3914 | .3914 | 0 | .175 |

**t = 3**

| | | x(1) | x(2) | x(3) | x(4) |
|---|---|---|---|---|---|
| $(i_3, j_3, s_3)$ | | (1,0,1) | | | |
| CONSTRAINTS$_3$ | | {x(3)≠1, x(1)=0} | | | |
| SCORE$_3$ | 0 | _**.175**_ | .175 | .175 | .0348 |
| | 1 | 0 | .024 | 0 | .175 |
| $x_3$ | | 0 | 0 | 0 | 1 |
| USED$_4$ | | {(3,1,1), (1,0,1)} | | | |
| CONSTRAINTS$_1$ | | {x(3)≠1 , x(1)≠0} | | | |
| SCORE$_1$ | 0 | 0 | .0145 | .3914 | .3914 |
| | 1 | .3914 | .3914 | 0 | .0783 |
| CONSTRAINTS$_2$ | | {x(3)=1} | | | |
| SCORE$_2$ | 0 | .035 | .035 | 0 | .1892 |
| | 1 | .1892 | .1892 | _**.1892**_ | .035 |

Figure 2: An illustration of our novel BMMF algorithm on a simple example.

most probable candidate out of these $k(N-1)+1$ is guaranteed to be $m_k$.

As pointed out by Nilsson, this simple algorithm may require far too many calculations of the MMs ($O(MN)$). He suggested an algorithm that uses traceback operations to reduce the computation significantly. Since traceback operations are problematic in loopy graphs, we now present a novel algorithm that does not use traceback but may require far less calculation of the MMs compared to SMFP.

## 2.2 A novel algorithm: Best Max-Marginal First

For simplicity of exposition, we will describe the BMMF algorithm under what we call the *strict order assumption*, that no two configurations have exactly the same probability.

We illustrate our algorithm using a simple example (figure 2). There are 4 binary variables in the graphical model and we can find the top 3 MPCs exactly: $1100, 1110, 0001$.

Our algorithm outputs a set of candidates $x_t$, one at each iteration. In the first iteration, $t = 1$, we start by calculating the MMs, and using the max-marginal lemma we find $m_1$. We now search the max-marginal table for the *next best* max-marginal value. In this case it is obtained with $x(3) = 1$. In the second iteration, $t = 2$, we now *lock* $x(3) = 1$. In other words, we calculate the MMs with the added constraint that $x(3)$ to 1. We use the max-marginal lemma to find the most likely configuration with $x(3) = 1$ locked and obtain $x_2 = 1110$. Note that we have found the second most likely configuration. We then add the complementary constraint $x(3) \neq 1$ to the originating constraints set and calculate the MMs. In the third iteration, $t = 3$, we search both previous max-marginal tables and find the best remaining max-marginal. It is obtained at $x(1) = 0$, $t = 1$. We now add the constraint $x(1) = 0$ to the constraints set from $t = 1$, calculate the MMs and use the max-marginal lemma to find $x_3 = 0001$. Finally, we add the complementary constraint $x(1) \neq 0$ to the originating constraints set and calculate the MMs. Thus

after 3 iterations we have found the first 3 MPCs using only 5 calculations of the MMs.

**The Best Max-Marginal First (BMMF) algorithm for calculating the $M$ most probable configurations:**

- *Initialization*

$$\mathsf{SCORE}_1(i,j) = \max_{x:x(i)=j} \Pr(X=x|y) \tag{2}$$

$$x_1(i) = \arg\max_j \mathsf{SCORE}_1(i,j) \tag{3}$$

$$\mathsf{CONSTRAINTS}_1 = \emptyset \tag{4}$$

$$\mathsf{USED}_2 = \emptyset \tag{5}$$

- *For t=2:T*

$$\mathsf{SEARCH}_t = (i,j,s<t:x_s(i) \neq j,(i,j,s) \notin \mathsf{USED}_t) \tag{6}$$

$$(i_t,j_t,s_t) = \arg\max_{(i,j,s)\in\mathsf{SEARCH}_t} \mathsf{SCORE}_s(i,j) \tag{7}$$

$$\mathsf{CONSTRAINTS}_t = \mathsf{CONSTRAINTS}_{s_t} \cup \{(\mathsf{x}(i_t)=j_t)\} \tag{8}$$

$$\mathsf{SCORE}_t(i,j) = \max_{x:x(i)=j,\mathsf{CONSTRAINTS}_t} \Pr(X=x|y) \tag{9}$$

$$x_t(i) = \arg\max_j \mathsf{SCORE}_t(i,j) \tag{10}$$

$$\mathsf{USED}_{t+1} = \mathsf{USED}_t \cup \{(i_t,j_t,s_t)\} \tag{11}$$

$$\mathsf{CONSTRAINTS}_{s_t} = \mathsf{CONSTRAINTS}_{s_t} \cup \{(\mathsf{x}(i_t) \neq j_t)\} \tag{12}$$

$$\mathsf{SCORE}_{s_t}(i,j) = \max_{x:x(i)=j,\mathsf{CONSTRAINTS}_{s_t}} \Pr(X=x|y) \tag{13}$$

*Claim 1:* $x_1$ calculated by the BMMF algorithm is equal to the MPC $m_1$.

*Proof:* This is just a restatement of the max-marginal lemma.

*Claim 2:* $x_2$ calculated by the BMMF algorithm is equal to the second MPC $m_2$.

*Proof:* We first show that $m_2(i_2) = j_2$. We know that $m_2$ differs in at least one location from $m_1$. We also know that out of all the assignments that differ from $m_1$ it must have the highest probability. Suppose, that $m_2(i_2) \neq j_2$. By the definition of $\mathsf{SCORE}_1$, this means that there exists an $x \neq m_2$ that is not $m_1$ whose posterior probability is higher than that of $m_2$. This is a contradiction. Now, out of all assignments for which $x(i_2) = j_2$, $m_2$ has highest posterior probability (recall that by definition, $m_1(i_2) \neq j_2$). The max-marginal lemma guarantees that $x_2 = m_2$. ∎

*Partition Lemma:* Let $\mathsf{SAT}_k$ denote the set of assignments satisfying $\mathsf{CONSTRAINTS}_k$. Then, after iteration $k$, the collection $\{\mathsf{SAT}_1, \mathsf{SAT}_2, \cdots, \mathsf{SAT}_k\}$ is a partition of the assignment space.

*Proof:* By induction over $k$. For $k = 1$, $\mathsf{CONSTRAINTS}_1 = \emptyset$ and the claim trivially holds. For $k = 2$, $\mathsf{SAT}_1 = \{\mathsf{x}|\mathsf{x}(i_2) \neq j_2\}$ and $\mathsf{SAT}_2 = \{\mathsf{x}|\mathsf{x}(i_2) = j_2\}$ are mutually disjoint and $\mathsf{SAT}_1 \cup \mathsf{SAT}_2$ covers the assignment space, therefore $\{\mathsf{SAT}_1, \mathsf{SAT}_2\}$ is a partition of the assignment space. Assume that after iteration $k-1$, $\{\mathsf{SAT}_1, \mathsf{SAT}_2, \cdots, \mathsf{SAT}_{k-1}\}$ is a partition of the assignment space. Note that in iteration $k$, we add $\mathsf{CONSTRAINTS}_k = \mathsf{CONSTRAINTS}_{s_k} \cup \{(\mathsf{x}(i_k) = j_k)\}$ and modify $\mathsf{CONSTRAINTS}_{s_k} = \mathsf{CONSTRAINTS}_{s_k} \cup \{(\mathsf{x}(i_k) \neq j_k)\}$, while keeping all other constraints set unchanged. $\mathsf{SAT}_k$ and the modified $\mathsf{SAT}_{s_k}$ are pairwise disjoint and $\mathsf{SAT}_k \cup \mathsf{SAT}_{s_k}$ covers the originating $\mathsf{SAT}_{s_k}$. Since after itera-

tion $k - 1$ $\{\mathsf{SAT}_1, \mathsf{SAT}_2, \cdots, \mathsf{SAT}_{k-1}\}$ is a partition of the assignment space, so is $\{\mathsf{SAT}_1, \mathsf{SAT}_2, \cdots, \mathsf{SAT}_k\}$. □

*Claim 3:* $x_k$, the configuration calculated by the algorithm in iteration $k$, is $m_k$, the $k$th MPC.

*Proof:* First, note that $\mathsf{SCORE}_{\mathsf{s}_k}(\mathsf{i}_k, \mathsf{j}_k) \leq \mathsf{SCORE}_{\mathsf{s}_{k-1}}(\mathsf{i}_{k-1}, \mathsf{j}_{k-1})$, otherwise $(i_k, j_k, s_k)$ would have been chosen in iteration $k - 1$. Following the partition lemma, each assignment arises at most once. By the strict order assumption, this means that $\mathsf{SCORE}_{\mathsf{s}_k}(\mathsf{i}_k, \mathsf{j}_k) < \mathsf{SCORE}_{\mathsf{s}_{k-1}}(\mathsf{i}_{k-1}, \mathsf{j}_{k-1})$.

Let $m_k \in \mathsf{SAT}_{\mathsf{s}^*}$. We know that $m_k$ differs from all previous $x_s$ in at least one location. In particular, $m_k$ must differ from $x_{s^*}$ in at least one location. Denote that location by $i^*$ and $m_k(i^*) = j^*$. We want to show that $\mathsf{SCORE}_{\mathsf{s}^*}(\mathsf{i}^*, \mathsf{j}^*) = \Pr(X = m_k | y)$. First, note that $(i^*, j^*, s^*) \notin \mathsf{USED}_k$. If we had previously used it, then $(x(i^*) \neq j^*) \in \mathsf{CONSTRAINTS}_{\mathsf{s}^*}$, which contradicts the definition of $s^*$. Now suppose there exists $m_l$, $l \leq k - 1$ such that $m_l \in \mathsf{SAT}_{\mathsf{s}^*}$ and $m_l(i^*) = j^*$. Since $(i^*, j^*, s^*) \notin \mathsf{USED}_k$ this would mean that $\mathsf{SCORE}_{\mathsf{s}_k}(\mathsf{i}_k, \mathsf{j}_k) \geq \mathsf{SCORE}_{\mathsf{s}_{k-1}}(\mathsf{i}_{k-1}, \mathsf{j}_{k-1})$ which is a contradiction. Therefore $m_k$ is the most probable assignment that satisfies $\mathsf{CONSTRAINTS}_{\mathsf{s}^*}$ and has the value $j^*$ at location $i^*$. Hence $\mathsf{SCORE}_{\mathsf{s}^*}(\mathsf{i}^*, \mathsf{j}^*) = \Pr(X = m_k | y)$. □

A consequence of claim 3 is that *BMMF will find the top M MPCs using* $2M$ *calculations of max marginals*. In contrast, SMFP requires $O(MN)$ calculations. In real world loopy problems, especially when $N \gg M$, this can lead to drastically different run times. First, real world problems may have thousands of nodes so a speedup of a factor of $N$ will be very significant. Second, calculating the MMs requires iterative algorithms (e.g. BP or GBP) so that the speedup of a factor of $N$ may be the difference between running a month versus running half a day.

# 3 Approximate MPCs algorithms using loopy BP

We now compare 4 approximate MPCs algorithms:

1. **loopy BMMF**. This is exactly the algorithm in section 2.2 with the MMs based on the beliefs computed by loopy max-product BP or max-GBP:

$$\mathsf{SCORE}_k(\mathsf{i}, \mathsf{j}) = \Pr(X = x_k | y) \frac{\mathsf{BEL}(\mathsf{i}, \mathsf{j} | \mathsf{CONSTRAINTS}_k)}{\max_{\mathsf{j}} \mathsf{BEL}(\mathsf{i}, \mathsf{j} | \mathsf{CONSTRAINTS}_k)} \qquad (14)$$

2. **loopy SMFP**. This is just Nilsson's SMFP algorithm with the MMs calculated using loopy max-product BP.

3. **Gibbs sampling**. We collect all configurations sampled during a Gibbs sampling simulation and output the top $M$ of these.

4. **Greedy**. We collect all configurations encountered during a greedy optimization of the posterior probability (this is just Gibbs sampling at zero temperature) and output the top $M$ of these.

All four algorithms were implemented in Matlab and the number of iterations for greedy and Gibbs were chosen so that the run times would be the same as that of loopy BMMF. Gibbs sampling started from $m_1$, the most probable assignment, and the greedy local search algorithm initialized to an assignment "similar" to $m_1$ (1% of the variables were chosen randomly and their values flipped).

For the protein folding problem [17], we used a database consisting of 325 proteins, each gives rise to a graphical model with hundreds of variables and many loops. We

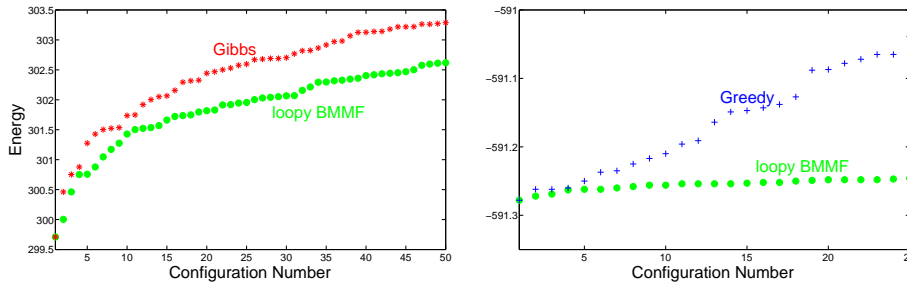

Figure 3: The configurations found by loopy-BMMF compared to those obtained using Gibbs sampling and greedy local search for a large toy-QMR model (right) and a $32 \times 32$ spin glass model (right).

compared the top 100 correct configurations obtained by the $A^*$ heuristic search algorithm [8] to those found by loopy BMMF algorithm, using BP. In all cases where $A^*$ was feasible, loopy BMMF always found the correct configurations. Also, the BMMF algorithm converged more often (96.3% compared to 76.3%) and ran much faster.

We then assessed the performance of the BMMF algorithm for a couple of relatively small problems, where exact inference was possible. For both a small toy-QMR model (with 20 diseases and 50 symptoms) and a $8 \times 8$ spin glass model the BMMF algorithm obtained the correct MPCs.

Finally, we compared the performance of the algorithms for couple of hard problems — a large toy-QMR model (with 100 diseases and 200 symptoms) and $32 \times 32$ spin glass model with large pairwise interactions. For the toy-QMR model, the MPCs calculated by the BMMF algorithm were better than those calculated by Gibbs sampling (Figure 3, left). For the large spin glass, we found that ordinary BP didn't converge and used max-product generalized BP instead. This is exactly the algorithm described in [18] with marginalizations replaced with maximizations. We found that GBP converged far more frequently and indeed the MPCs found using GBP are much better than those obtained with Gibbs or greedy (Figure 3, right. Gibbs results are worse than those of the greedy search and therefore not shown). Note that finding the second MPC using the simple MFP algorithm requires a week, while the loopy BMMF calculated the 25 MPCs in few hours only.

## 4    Discussion

Existing algorithms successfully find the $M$ MPCs for graphs where building a JT is possible. However, in many real-world applications exact inference is impossible and approximate techniques are needed. In this paper we have addressed the problem of finding the $M$ MPCs using the techniques of approximate inference. We have presented a new algorithm, called Best Max-Marginal First that will provably solve the problem if MMs can be calculated exactly. We have shown that the algorithm continues to perform well when the MMs are approximated using max-product loopy BP or GBP.

Interestingly, the BMMF algorithm uses the numerical values of the approximate MMs to determine what to do in each iteration. The success of loopy BMMF suggests that in some cases the max product loopy BP gives a good numerical approximation to the true MMs. Most existing analysis of loopy max-product [16,

15] has focused on the configurations found by the algorithm. It would be interesting to extend the analysis to bound the approximate MMs which in turn would lead to a provable approximate MPCs algorithm.

While we have used loopy BP to approximate the MMs, any approximate inference can be used inside BMMF to derive a novel, approximate MPCs algorithm. In particular, the algorithm suggested by Wainwright et al. [14] can be shown to give the MAP assignment when it converges. It would be interesting to incorporate their algorithm into BMMF.

# References

[1] A. Cano, S. Moral, and A. Salmerón. Penniless propagation in join trees. *Journal of Intelligent Systems*, 15:1010–1027, 2000.

[2] R. Cowell. Advanced inference in Bayesian networks. In M.I. Jordan, editor, *Learning in Graphical Models*. MIT Press, 1998.

[3] P. Dawid. Applications of a general propagation algorithm for probabilistic expert systems. *Statistics and Computing*, 2:25–36, 1992.

[4] R. Dechter and I. Rish. A scheme for approximating probabilistic inference. In *Uncertainty in Artificial Intelligence (UAI 97)*, 1997.

[5] A. Doucet, N. de Freitas, K. Murphy, and S. Russell. Rao-blackwellised particle filtering for dynamic bayesian networks. In *Proceedings UAI 2000*. Morgan Kaufmann, 2000.

[6] B.J. Frey, R. Koetter, and N. Petrovic. Very loopy belief propagation for unwrapping phase images. In *Adv. Neural Information Processing Systems 14*. MIT Press, 2001.

[7] T.S. Jaakkola and M.I. Jordan. Variational probabilistic inference and the QMR-DT database. *JAIR*, 10:291–322, 1999.

[8] Andrew R. Leach and Andrew P. Lemon. Exploring the conformational space of protein side chains using dead-end elimination and the A* algorithm. *Proteins: Structure, Function, and Genetics*, 33(2):227–239, 1998.

[9] A. Levin, A. Zomet, and Y. Weiss. Learning to perceive transparency from the statistics of natural scenes. In *Proceedings NIPS 2002*. MIT Press, 2002.

[10] Kevin Murphy. The bayes net toolbox for matlab. *Computing Science and Statistics*, 33, 2001.

[11] D. Nilsson. An efficient algorithm for finding the M most probable configurations in probabilistic expert systems. *Statistics and Computing*, 8:159–173, 1998.

[12] Judea Pearl. *Probabilistic Reasoning in Intelligent Systems: Networks of Plausible Inference*. Morgan Kaufmann, 1988.

[13] L.R. Rabiner. A tutorial on hidden Markov models and selected applications in speech recognition. *Proc. IEEE*, 77(2):257–286, 1989.

[14] M. J. Wainwright, T. Jaakkola, and A. S. Willsky. Exact map estimates by (hyper)tree agreement. In *Proceedings NIPS 2002*. MIT Press, 2002.

[15] M. J. Wainwright, T. Jaakkola, and A. S. Willsky. Tree consistency and bounds on the performance of the max-product algorithm and its generalizations. Technical Report P-2554, MIT LIDS, 2002.

[16] Y. Weiss and W.T. Freeman. On the optimality of solutions of the max-product belief propagation algorithm in arbitrary graphs. *IEEE Transactions on Information Theory*, 47(2):723–735, 2001.

[17] C. Yanover and Y. Weiss. Approximate inference and protein folding. In *Proceedings NIPS 2002*. MIT Press, 2002.

[18] J. Yedidia, W. Freeman, and Y. Weiss. Understanding belief propagation and its generalizations. In G. Lakemeyer and B. Nebel, editors, *Exploring Artificial Intelligence in the New Millennium*. Morgan Kaufmann, 2003.
